# INTRODUCTION TO A SYSTEM FOR IMPLEMENTING NEURAL NET CONNECTIONS ON SIMD ARCHITECTURES

Sherryl Tomboulian

Institute for Computer Applications in Science and Engineering
NASA Langley Research Center, Hampton VA 23665

## ABSTRACT

Neural networks have attracted much interest recently, and using parallel architectures to simulate neural networks is a natural and necessary application. The SIMD model of parallel computation is chosen, because systems of this type can be built with large numbers of processing elements. However, such systems are not naturally suited to generalized communication. A method is proposed that allows an implementation of neural network connections on massively parallel SIMD architectures. The key to this system is an algorithm that allows the formation of arbitrary connections between the "neurons". A feature is the ability to add new connections quickly. It also has error recovery ability and is robust over a variety of network topologies. Simulations of the general connection system, and its implementation on the Connection Machine, indicate that the time and space requirements are proportional to the product of the average number of connections per neuron and the diameter of the interconnection network.

## INTRODUCTION

Neural Networks hold great promise for biological research, artificial intelligence, and even as general computational devices. However, to study systems in a realistic manner, it is highly desirable to be able to simulate a network with tens of thousands or hundreds of thousands of neurons. This suggests the use of parallel hardware. The most natural method of exploiting parallelism would have each processor simulating a single neuron.

Consider the requirements of such a system. There should be a very large number of processing elements which can work in parallel. The computation that occurs at these elements is simple and based on local data. The processing elements must be able to have connections to other elements. All connections in the system must be able to be traversed in parallel. Connections must be added and deleted dynamically.

Given current technology, the only type of parallel model that can be constructed with tens of thousands or hundreds of thousands of processors is an SIMD architecture. In exchange for being able to build a system with so many processors, there are some inherent limitations. SIMD stands for single instruction multiple data[1] which means that all processors can work in parallel, but they must do exactly the same thing at the same time. This machine model is sufficient for the computation required within a neuron, however in such a system it is difficult to implement arbitrary connections between neurons. The Connection Machine[2] provides such a model, but uses a device called the router

This work was supported by the National Aeronautics and Space Administration under NASA Constract No. NAS1-18010-7 while the author was in residence at ICASE.

to deliver messages. The router is a complex piece of hardware that uses significant chip area, and without the additional hardware for the router, a machine could be built with significantly more processors. Since one of the objectives is to maximize the number of "neurons" it is desirable to eliminate the extra cost of a hardware router and instead use a software method.

Existing software algorithms for forming connections on SIMD machines are not sufficient for the requirements of a neural networks. They restrict the form of graph (neural network) that can be embedded to permutations[3,4] or sorts[5,6combinedwith7], the methods are network specific, and adding a new connection is highly time consuming.

The software routing method presented here is a unique algorithm which allows arbitrary neural networks to be embedded in machines with a wide variety of network topologies. The advantages of such an approach are numerous: A new connection can be added dynamically in the same amount of time that it takes to perform a parallel traversal of all connections. The method has error recovery ability in case of network failures. This method has relationships with natural neural models. When a new connection is to be formed, the two neurons being connected are activated, and then the system forms the connection without any knowledge of the "address" of the neuron-processors and without any instruction as to the method of forming the connecting path. The connections are entirely distributed; a processor only knows that connections pass through it – it doesn't know a connection's origin or final destination.

Some neural network applications have been implemented on massively parallel architectures, but they have run into restrictions due to communication. An implementation on the Connection Machine[8] discovered that it was more desirable to cluster processors in groups, and have each processor in a group represent one connection, rather than having one processor per neuron, because the router is designed to deliver one message at a time from each processor. This approach is contrary with the more natural paradigm of having one processor represent a neuron. The MPP [9], a massively parallel architecture with processors arranged in a mesh, has been used to implement neural nets[10], but because of a lack of generalized communication software, the method for edge connections is a regular communication pattern with all neurons within a specified distance. This is not an unreasonable approach, since within the brain neurons are usually locally connected, but there is also a need for longer connections between groups of neurons. The algorithms presented here can be used on both machines to facilitate arbitrary connections with an irregular number of connections at each processor.

## MACHINE MODEL

As mentioned previously, since we desire to build a system with an large number of processing elements, the only technology currently available for building such large systems is the SIMD architecture model. In the SIMD model there is a single control unit and a very large number of slave processors that can execute the same instruction stream simultaneously. It is possible to disable some processors so that only some execute an instruction, but it is not possible to have two processor performing different instructions at the same time. The processors have exclusively local memory which is small (only a few thousand bits), and they have no facilities for local indirect addressing. In this scheme an *instruction* involves both a particular operation code *and* the local memory

address. All processors must do this same thing to the same areas of their local memory at the same time.

The basic model of computation is bit-serial – each instruction operates on a bit at a time. To perform multiple bit operations, such as integer addition, requires several instructions. This model is chosen because it requires less hardware logic, and so would allow a machine to be built with a larger number of processors than could otherwise be achieved with a standard word-oriented approach. Of course, the algorithms presented here will also work for machines with more complex instruction abilities; the machine model described satisfies the minimal requirements.

An important requirement for connection formation is that the processors are connected in some topology. For instance, the processors might be connected in a grid so that each processor has a North, South, East, and West neighbor. The methods presented here work for a wide variety of network topologies. The requirements are: (1) there must be some path between any two processors; (2) every neighbor link must be bi-directional, i.e. if A is a neighbor of B, then B must be a neighbor of A; (3) the neighbor relations between processors must have a consistent invertible labeling. A more precise definition of the labeling requirements can be found in [11]. It suffices that most networks [12], including grid, hypercube, cube connected cycles[13], shuffle exchange[14], and mesh of trees[15] are admissible under the scheme. Additional requirements are that the processors be able to read from or write to their neighbors' memories, and that at least one of the processors acts as a serial port between the processors and the controller.

## COMPUTATIONAL REQUIREMENTS

The machine model described here is sufficient for the computational requirements of a neuron. Adopt the paradigm that each processor represents one neuron. While several different models of neural networks exist with slightly different features, they are all fairly well characterized by computing a sum or product of the neighbors values, and if a certain threshold is exceeded, then the processor neuron will *fire*, i.e. activate other neurons. The machine model described here is more efficient at boolean computation, such as described by McCulloch and Pitts[16], since it is bit serial. Neural net models using integers and floating point arithmetic [17,18] will also work but will be somewhat slower since the time for computation is proportional to the number of bits of the operands.

The only computational difficulty lies in the fact that the system is SIMD, which means that the processes are synchronous. For some neural net models this is sufficient[18] however others require asynchronous behavior [17]. This can easily be achieved simply by turning the processors on and off based on a specified probability distribution. (For a survey of some different neural networks see [19]).

## CONNECTION ASSUMPTIONS

Many models of neural networks assume fully connected systems. This model is considered unrealistic, and the method presented here will work better for models that contain more sparsely connected systems. While the method will work for dense connections, the time and space required is proportional to

the number of edges, and becomes prohibitively expensive.

Other than the sparse assumptions, there are no restrictions to the topological form of the network being simulated. For example, multiple layered systems, slightly irregular structures, and completely random connections are all handled easily. The system does function better if there is locality in the neural network. These assumptions seem to fit the biological model of neurons.

## THE CONNECTION FORMATION METHOD

A fundamental part of a neural network implementation is the realization of the connections between neurons. This is done using a software scheme first presented in [11,20]. The original method was intended for realizing directed graphs in SIMD architectures. Since a neural network is a graph with the neurons being vertices and the connections being arcs, the method maps perfectly to this system. Henceforth the terms neuron and vertex and the terms arc and connection will be used interchangeably.

The software system presented here for implementing the connections has several parts. Each processor will be assigned exactly one neuron. (Of course some processors may be "free" or unallocated, but even "free" processor participate in the routing process.) Each connection will be realized as a path in the topology of processors. A labeling of these paths in time and space is introduced which allows efficient routing algorithms and a set-up strategy is introduced that allows new connections to be added quickly.

The standard computer science approach to forming the connection would be to store the addresses of the processors to which a given neuron is connected. Then, using a routing algorithm, messages could be passed to the processors with the specified destination. However, the SIMD architecture does not lend itself to standard message passing schemes because processors cannot do indirect addressing, so buffering of values is difficult and costly.

Instead, a scheme is introduced which is closer to the natural neuron-synapse structures. Instead of having an address for each connection, the connection is actually represented as a fixed path between the processors, using time as a virtual dimension. The path a connection takes through the network of processors is statically encoded in the local memories of the neurons that it passes through. To achieve this, the following data structures will be resident at each processor.

```
ALLOCATED ---- boolean flag indicating
    whether this processor is assigned
    a vertex (neuron) in the graph
VERTEX LABEL --- label of graph vertex  (neuron)
HAS_NEIGHBOR[1..neighbor_limit] flag
    indicating the existence of neighbors
SLOTS[1..T] OF      arc path information
    START----------new arc starts here
    DIRECTION------direction to send
                  {1..neighbor_limit,FREE}
    END----------arc ends here
    ARC LABEL-----label of arc
```

The ALLOCATED and VERTEX LABEL field indicates that the processor has been assigned a vertex in the graph (neuron). The HAS NEIGHBOR field is used to indicate whether a physical wire exists in the particular direction; it allows irregular network topologies and boundary conditions to be supported. The SLOTS data structure is the key to realizing the connections. It is used to instruct the processor where to send a message and to insure that paths are constructed in such a way that no collisions will occur.

SLOTS is an array with T elements. The value T is called the time quantum. Traversing all the edges of the embedded graph in parallel will take a certain amount of time since messages must be passed along through a sequence of neighboring processors. Forming these parallel connections will be considered an uninterruptable operation which will take T steps. The SLOTS array is used to tell the processors what they should do on each relative time position within the time quantum.

One of the characteristics of this algorithm is that a fixed path is chosen to represent the connection between two processors, and once chosen it is never changed. For example, consider the grid below.

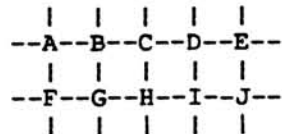

Fig. 1. Grid Example

If there is an arc between A and H, there are several possible paths: East-East-South, East-South-East, and South-East-East. Only one of these paths will be chosen between A and H, and that same path will always be used. Besides being invariant in space, paths are also invariant in time. As stated above, traversal is done within a time quantum T. Paths do no have to start on time 1, but can be scheduled to start at some relative offset within the time quantum. Once the starting time for the path has been fixed, it is never changed. Another requirement is that a message can not be buffered, it must proceed along the specified directions without interruption. For example, if the path is of length 3 and it starts at time 1, then it will arrive at time 4. Alternatively, if it starts at time 2 it will arrive at time 5. Further, it is necessary to place the paths so that no collisions occur; that is, no two paths can be at the same processor at the same instant in time. Essentially time adds an extra dimension to the topology of the network, and within this space-time network all data paths must be non-conflicting. The rules for constructing paths that fulfill these requirements are listed below.

- At most one connection can enter a processor at a given time, and at most one connection can leave a processor at a given time. It is possible to have both one coming and one going at the same time. Note that this does not mean that a processor can have only one connection; it means that it can have only one connection during any one of the T time steps. It can have as many as T connections going through it.

- Any path between two processors (u,v) representing a connection must consist of steps at contiguous times. For example, if the path from processor u to processor v is u,f,g,h,v , then if the arc from u-f is assigned time 1, f-g must have time 2, g-h time 3, and h-v time 4. Likewise if u-f occurs at time 5, then arc h-v will occur time 8.

When these rules are used when forming paths, the SLOTS structure can be used to mark the paths. Each path goes through neighboring processors at successive time steps. For each of these time steps the DIRECTION field of the SLOTS structure is marked, telling the processor which direction it should pass a message if it receives it on that time. SLOTS serves both to instruct the processors how to send messages, and to indicate that a processor is busy at a certain time slot so that when new paths are constructed it can be guaranteed that they won't conflict with current paths.

Consider the following example. Suppose we are given the directed graph with vertices A,B,C,D and edges $A - > C$, $B - > C, B - > D$, and $D - > A$. This is to be done where A,B,C, and D have been assigned to successive elements of a linear array. ( A linear array in not a good network for this scheme, but is a convenient source of examples.)

Logical Connections

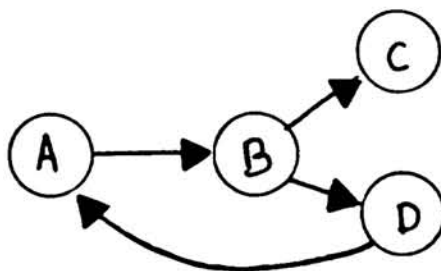

Fig. 2. Graph Example

A,B,C,D are successive members in a linear array

```
1---2---3---4
A---B---C---D
```

First, A ->C can be completed with the map East-East, so
Slots[A][1].direction  = E, Slots[B][2].direction=E,
Slots[C][2].end = 1 .

B->C can be done with the map East, it can start at time 1,
since Slots[B][1].direction and Slots[C][1].end are free.

B->D goes through C then to D, its map is East-East. B is
occupied at time 1 and 2.  It is free at time 3,
so Slots[B][3].direction = E, Slots[C][4].direction = E,
Slots[D][4].end = 1.

D->A must go through C,B,A. using map West-West-West.
D is free on time 1, C is free on time 2, but B is occupied
on time 3.  D is free on time 2, but C is occupied on time 3.
It can start from D at time 3, Slots[D][3].direction = W,
Slots[C][4].direction = W, Slots[B][5].direction = W,
Slots[A][5].end=1

Every processor acts as a conduit for its neighbors messages. No processor knows where any message is going to or coming from, but each processor knows what it must do to establish the local connections.

The use of contiguous time slots is vital to the correct operation of the system. If all edge-paths are established according to the above rules, there is a simple method for making the connections. The paths have been restricted so that there will be no collisions, and paths' directions use consecutive time slots. Hence if all arcs at time $i$ send a message to their neighbors, then each processor is guaranteed no more than 1 message coming to it. The end of a path is specified by setting a separate bit that is tested after each message is received. A separate start bit indicates when a path starts. The start bit is needed because the SLOTS array just tells the processors where to send a message, regardless of how that message arrived. The start array indicates when a message originates, as opposed to arriving from a neighbor.

The following algorithm is basic to the routing system.

```
for i = time 1 to T
        FORALL processors
    /* if an arc starts or is passing through at this time*/
            if SLOT[i].START = 1 or active = 1
                for j=1 to neighbor-limit
                    if SLOT[i].direction= j
                        write message bit to in-box
                            of neighbor j;
            set active = 0;
            FORALL processor that just received a message
            if end[i]
                move in-box to message-destination;
            else
                move in-box to out-box;
                set active bit = 1;
```

This code follows the method mentioned above. The time slots are looped through and the messages are passed in the appropriate directions as specified in the SLOTS array. Two bits, in-box and out-box, are used for message passing so that an out-going message won't be overwritten by an in-coming message before it gets transferred. The inner loop *for j = 1 to neighbor limit* checks each of the possible neighbor directions and sends the message to the correct neighbor. For instance, in a grid the neighbor limit is 4, for North, South, East, and West neighbors. The time complexity of data movement is O(T times neighbor-limit).

## SETTING UP CONNECTIONS

One of the goals in developing this system was to have a method for adding new connections quickly. Paths are added so that they don't conflict with any previously constructed path. Once a path is placed it will not be re-routed

by the basic placement algorithm; it will always start at the same spot at the same time. The basic idea of the method for placing a connection is to start from the source processor and in parallel examine all possible paths outward from it that do not conflict with pre-established paths and which adhere to the sequential time constraint. As the trial paths are flooding the system, they are recorded in temporary storage. At the end of this deluge of trial paths *all* possible paths will have been examined. If the destination processor has been reached, then a path exists under the current time-space restrictions. Using the stored information a path can be backtraced and recorded in the SLOTS structure. This is similar to the Lee-Moore routing algorithm[21,22] for finding a path in a system, but with the sequential time restriction.

For example, suppose that the connection (u,v) is to be added. First it is assumed that processors for u and v have already been determined, otherwise (as a simplification) assume a random allocation from a pool of free processors. A parallel breadth-first search will be performed starting from the source processor. During the propagation phase a processor which receives a message checks its SLOTS array to see if they are busy on that time step, if not it will propagate to its neighbors on the next time step. For instance, suppose a trial path starts at time 1 and moves to a neighboring processor, but that neighbor is already busy at time 1 (as can be seen by examining the DIRECTION-SLOT.) Since a path that would go through this neighbor at this time is not legal, the trial path would commit suicide, that is, it stops propagating itself. If the processor slot for time 2 was free, the trial path would attempt to propagate to all of its' neighbors at time 3.

Using this technique paths can be constructed with essentially no knowledge of the relative locations of the "neurons" being connected or the underlying topology. Variations on the outlined method, such as choosing the shortest path, can improve the choice of paths with very little overhead. If the entire network were known ahead of time, an off-line method could be used to construct the paths more efficiently; work on off-line methods is underway. However, the simple elegance of this basic method holds great appeal for systems that change slowly over time in unpredictable ways.

## PERFORMANCE

Adding an edge (assuming one can be added), deleting any set of edges, or traversing all the edges in parallel, all have time complexity $O(T \times neighbor - limit)$. If it is assumed that neighbor limit is a small constant then the complexity is $O(T)$. Since T is related both to the time and space needed, it is a crucial factor in determining the value of the algorithms presented. Some analytic bounds on T were presented in[11], but it is difficult to get a tight bound on T for general interconnection networks and dynamically changing graphs. A simulator was constructed to examine the behavior of the algorithms. Besides the simulated data, the algorithms mentioned were actually implemented for the Connection Machine. The data produced by the simulator is consistent with that produced by the real machine. The major result is that the size of T appears proportional to the average degree of the graph times the diameter of the interconnection network[20].

## FURTHER RESEARCH

This paper has been largely concerned with a system that can realize the connections in a neural network when the two neurons to be joined have been activated. The tests conducted have been concerned with the validity of the method for implementing connections, rather than with a full simulation of a neural network. Clearly this is the next step.

A natural extension of this method is a system which can form its own connections based solely on the activity of certain neurons, without having to explicitly activate the source and destination neurons. This is an exciting avenue, and further results should be forthcoming.

Another area of research involves the formation of branching paths. The current method takes an arc in the neural network and realizes it as a unique path in space-time. A variation that has similarities to dendritic structure would allow a path coming from a neuron to branch and go to several target neurons. This extension would allow for a much more economical embedding system. Simulations are currently underway.

## CONCLUSIONS

A method has been outlined which allows the implementation of neural nets connections on a class of parallel architectures which can be constructed with very large numbers of processing elements. To economize on hardware so as to maximize the number of processing element buildable, it was assumed that the processors only have local connections; no hardware is provided for communication. Some simple algorithms have been presented which allow neural nets with arbitrary connections to be embedded in SIMD architectures having a variety of topologies. The time for performing a parallel traversal and for adding a new connection appears to be proportional to the diameter of the topology times the average number of arcs in the graph being embedded. In a system where the topology has diameter $O(logN)$, and where the degree of the graph being embedded is bounded by a constant, the time is apparently $O(logN)$. This makes it competitive with existing methods for SIMD routing, with the advantages that there are no apriori requirements for the form of the data, and the topological requirements are extremely general. Also, with our approach new arcs can be added without reconfiguring the entire system. The simplicity of the implementation and the flexibility of the method suggest that it could be an important tool for using SIMD architectures for neural network simulation.

## BIBLIOGRAPHY

1. M.J. Flynn, "Some computer organizations and their effectiveness", IEEE Trans Comput., vol C-21, no.9, pp. 948-960.
2. W. Hillis, "The Connection Machine", MIT Press, Cambridge, Mass, 1985.
3. D. Nassimi, S. Sahni, "Parallel Algorithms to Set-up the Benes Permutation Network", Proc. Workshop on Interconnection Networks for Parallel and Distributed Processing, April 1980.
4. D. Nassimi, S. Sahni, "Benes Network and Parallel Permutation Algorithms", IEEE Transactions on Computers, Vol C-30, No 5, May 1981.
5. D. Nassimi, S. Sahni, "Parallel Permutation and Sorting Algorithms and a

New Generalized Connection Network", *JACM*, Vol. 29, No. 3, July 1982 pp. 642-667

6. K.E. Batcher, "Sorting Networks and their Applications", The Proceedings of AFIPS 1968 SJCC, 1968, pp. 307-314.

7. C. Thompson, "Generalized connection networks for parallel processor inter-communication", *IEEE Tran. Computers*, Vol C, No 27, Dec 78, pp. 1119-1125.

8. Nathan H. Brown, Jr., "Neural Network Implementation Approaches for the Connection Machine", presented at the 1987 conference on Neural Information Processing Systems – Natural and Synthetic.

9. K.E. Batcher, "Design of a massively parallel processor", *IEEE Trans on Computers*, Sept 1980, pp. 836-840.

10. H.M. Hastings, S. Waner, "Neural Nets on the MPP", Frontiers of Massively Parallel Scientific Computation, NASA Conference Publication 2478, NASA Goddard Space Flight Center, Greenbelt Maryland, 1986.

11. S. Tomboulian, "A System for Routing Arbitrary Communication Graphs on SIMD Architectures", Doctoral Dissertation, Dept of Computer Science, Duke University, Durham NC.

12. T. Feng, "A Survey of Interconnection Networks", *Computer*, Dec 1981, pp.12-27.

13. F. Preparata and J. Vuillemin, "The Cube Connected Cycles: a Versatile Network for Parallel Computation", *Comm. ACM*, Vol 24, No 5 May 1981, pp. 300-309.

14. H. Stone, "Parallel processing with the perfect shuffle", *IEEE Trans. Computers*, Vol C, No 20, Feb 1971, pp. 153-161.

15. T. Leighton, "Parallel Computation Using Meshes of Trees", *Proc. International Workshop on Graph Theory Concepts in Computer Science*, 1983.

16. W.S. McCulloch, and W. Pitts, "A Logical Calculus of the Ideas Imminent in Nervous Activity," *Bulletin of Mathematical Biophysics*, Vol 5, 1943, pp.115-133.

17. J.J. Hopfield, "Neural networks and physical systems with emergent collective computational abilities", *Proc. Natl. Aca. Sci.*, Vol 79, April 1982, pp. 2554-2558.

18. T. Kohonen, "Self-Organization and Associative Memory, Springer-Verlag, Berlin , 1984.

19. R.P. Lippmann, "An Introduction to Computing with Neural Nets", *IEEE AASP*, April 1987, pp. 4-22.

20. S. Tomboulian, "A System for Routing Directed Graphs on SIMD Architectures", ICASE Report No. 87-14, NASA Langley Research Center, Hampton, VA.

21. C.Y. Lee, "An algorithm for path connections and its applications", *IRE Trans Elec Comput*, Vol. EC-10, Sept. 1961, pp. 346-365.

22. E. F. Moore, "Shortest path through a maze", *Annals of Computation Laboratory*, vol. 30. Cambridge, MA: Harvard Univ. Press, 1959, pp.285-292.